# Result Analysis of the NIPS 2003 Feature Selection Challenge

**Isabelle Guyon**
ClopiNet
Berkeley, CA 94708, USA
*isabelle@clopinet.com*

**Steve Gunn**
School of Electronics and Computer Science
University of Southampton, U.K.
*s.r.gunn@ecs.soton.ac.uk*

**Asa Ben Hur**
Department of Genome Sciences
University of Washington, USA
*asa@gs.washington.edu*

**Gideon Dror**
Department of Computer Science
Academic College of Tel-Aviv-Yaffo, Israel
*gideon@mta.ac.il*

## Abstract

The NIPS 2003 workshops included a feature selection competition organized by the authors. We provided participants with five datasets from different application domains and called for classification results using a minimal number of features. The competition took place over a period of 13 weeks and attracted 78 research groups. Participants were asked to make on-line submissions on the validation and test sets, with performance on the validation set being presented immediately to the participant and performance on the test set presented to the participants at the workshop. In total 1863 entries were made on the validation sets during the development period and 135 entries on all test sets for the final competition. The winners used a combination of Bayesian neural networks with ARD priors and Dirichlet diffusion trees. Other top entries used a variety of methods for feature selection, which combined filters and/or wrapper or embedded methods using Random Forests, kernel methods, or neural networks as a classification engine. The results of the benchmark (including the predictions made by the participants and the features they selected) and the scoring software are publicly available. The benchmark is available at `www.nipsfsc.ecs.soton.ac.uk` for post-challenge submissions to stimulate further research.

## 1 Introduction

Recently, the quality of research in Machine Learning has been raised by the sustained data sharing efforts of the community. Data repositories include the well known UCI Machine Learning repository [13], and dozens of other sites [10]. Yet, this has not diminished the importance of organized competitions. In fact, the proliferation of datasets combined with the creativity of researchers in designing

experiments makes it hardly possible to compare one paper with another [12]. A number of large conferences have regularly organized competitions (e.g. KDD, CAMDA, ICDAR, TREC, ICPR, and CASP). The NIPS workshops offer an ideal forum for organizing such competitions. In 2003, we organized a competition on the theme of feature selection, the results of which were presented at a workshop on feature extraction, which attracted 98 participants. We are presently preparing a book combining tutorial chapters and papers from the proceedings of that workshop [9]. In this paper, we present to the NIPS community a concise summary of our challenge design and the findings of the result analysis.

## 2 Benchmark design

We formatted five datasets (Table 1) from various application domains. All datasets are two-class classification problems. The data were split into three subsets: a training set, a validation set, and a test set. All three subsets were made available at the beginning of the benchmark, on September 8, 2003. The class labels for the validation set and the test set were withheld. The identity of the datasets and of the features (some of which were random features artificially generated) were kept secret. The participants could submit prediction results on the validation set and get their performance results and ranking on-line for a period of 12 weeks. By December $1^{st}$, 2003, which marked the end of the development period, the participants had to turn in their results on the test set. Immediately after that, the validation set labels were revealed. On December $8^{th}$, 2003, the participants could make submissions of test set predictions, after having trained on both the training and the validation set. Some details on the benchmark design are provided in this Section.

### Challenge format

We gave our benchmark the format of a challenge to stimulate participation. We made available an automatic web-based system to submit prediction results and get immediate feed-back, inspired by the system of the NIPS2000 and NIPS2001 unlabelled data competitions [4, 5]. However, unlike what had been done for these other competitions, we used a "validation set" to assess performance during the development period, and a separate "test set" for final scoring.

During development participants could submit validation results on any of the five datasets proposed (not necessarily all). Competitors were required to submit results on all five test sets by the challenge deadline to be included in the final ranking. This avoided a common problem of "multiple track" benchmarks in which no conclusion can be drawn because too few participants enter all tracks.

To promote collaboration between researchers, reduce the level of anxiety, and let people explore various strategies (e.g. "pure" methods and "hybrids"), we allowed participating groups to submit a total of five final entries on December $1^{st}$ and five entries on December $8^{th}$.

Our format was very successful: it attracted 78 research groups who competed for 13 weeks and made (submitted) a total of 1863 entries. Twenty groups were eligible for being ranked on December $1^{st}$ (56 submissions[1]), and 16 groups on December $8^{th}$ (36 submissions.) The feature selection benchmark web site at `www.nipsfsc.ecs.soton.ac.uk` remains available as a resource for researchers in the feature selection.

Table 1: **NIPS 2003 challenge datasets.** For each dataset we show the domain it was taken from, its type (dense, sparse, or sparse binary), the number of features, the percentage of probes, and the number of examples in the training, validation, and test sets. All problems are two-class classification problems.

| Dataset | Domain | Type | #Fe | %Pr | #Tr | #Val | #Te |
|---|---|---|---|---|---|---|---|
| ARCENE | Mass Spectrometry | Dense | 10000 | 30 | 100 | 100 | 700 |
| DEXTER | Text classification | Sparse | 20000 | 50 | 300 | 300 | 2000 |
| DOROTHEA | Drug discovery | Sparse binary | 100000 | 50 | 800 | 350 | 800 |
| GISETTE | Digit recognition | Dense | 5000 | 30 | 6000 | 1000 | 6500 |
| MADELON | Artificial | Dense | 500 | 96 | 2000 | 600 | 1800 |

**The challenge datasets**

Until the late 90s most published papers on feature selection considered datasets with less than 40 features[2] (see [1, 11] from a 1997 special issue on *relevance* for example). The situation has changed considerably in the past few years, and in the 2003 special issue we edited for JMLR including papers from the proceedings of the NIPS 2001 workshop [7], most papers explore domains with hundreds to tens of thousands of variables or features. The applications are driving this effort: bioinformatics, chemistry (drug design, cheminformatics), text processing, pattern recognition, speech processing, and machine vision provide machine learning problems in very high dimensional spaces, but often with comparably few examples.

Feature selection is a particular way of tackling the problem of space dimensionality reduction. The necessary computing power to handle large datasets is now available in simple laptops, so there is a proliferation of solutions proposed for such feature selection problems. Yet, there does not seem to be an emerging unity of experimental design and algorithms. We formatted five datasets for the purpose of benchmarking variable selection algorithms (see Table 1.)

The datasets were chosen to span a variety of domains and difficulties (the input variables are continuous or binary, sparse or dense; one dataset has unbalanced classes.) One dataset (MADELON) was artificially constructed to illustrate a particular difficulty: selecting a feature set when no feature is informative by itself. We chose datasets that had sufficiently many examples to create a large enough test set to obtain statistically significant results [6]. To prevent researchers familiar with the datasets to have an advantage, we concealed the identity of the datasets during the benchmark. We performed several preprocessing and data formatting steps, which contributed to disguising the origin of the datasets. In particular, we introduced a number of features called **probes**. The probes were drawn at random from a distribution resembling that of the real features, but carrying no information about the class labels. Such probes have a function in performance assessment: a good feature selection algorithm should eliminate most of the probes. The details of data preparation can be found in a technical memorandum [6].

Table 2: We show the top entries sorted by their score (times 100), the balanced error rate in percent (BER) and corresponding rank in parenthesis, the area under the ROC curve times 100 (AUC) and corresponding rank in parenthesis, the percentage of features used (Fe), and the percentage of probes in the features selected (Pr).

(a) **December $1^{st}$ 2003 challenge results.**

| Method (Team) | Score | BER | | AUC | | Fe | Pr |
|---|---|---|---|---|---|---|---|
| BayesNN-DFT (*Neal/Zhang*) | 88.0 | 6.84 | (1) | 97.22 | (1) | 80.3 | 47.8 |
| BayesNN-DFT (*Neal/Zhang*) | 86.2 | 6.87 | (2) | 97.21 | (2) | 80.3 | 47.8 |
| BayesNN-small (*Neal*) | 68.7 | 8.20 | (3) | 96.12 | (5) | 4.7 | 2.9 |
| BayesNN-large (*Neal*) | 59.6 | 8.21 | (4) | 96.36 | (3) | 60.3 | 28.5 |
| RF+RLSC (*Torkkola/Tuv*) | 59.3 | 9.07 | (7) | 90.93 | (29) | 22.5 | 17.5 |
| final2 (*Chen*) | 52.0 | 9.31 | (9) | 90.69 | (31) | 24.9 | 12.0 |
| SVMBased3 (*Zhili/Li*) | 41.8 | 9.21 | (8) | 93.60 | (16) | 29.5 | 21.7 |
| SVMBased4 (*Zhili/Li*) | 41.1 | 9.40 | (10) | 93.41 | (18) | 29.5 | 21.7 |
| final1 (*Chen*) | 40.4 | 10.38 | (23) | 89.62 | (34) | 6.2 | 6.1 |
| transSVM2 (*Zhili*) | 36.0 | 9.60 | (13) | 93.21 | (20) | 29.5 | 21.7 |
| BayesNN-E (*Neal*) | 29.5 | 8.43 | (5) | 96.30 | (4) | 96.8 | 56.7 |
| Collection2 (*Saffari*) | 28.0 | 10.03 | (20) | 89.97 | (32) | 7.7 | 10.6 |
| Collection1 (*Saffari*) | 20.7 | 10.06 | (21) | 89.94 | (33) | 32.3 | 25.5 |

(b) **December $8^{th}$ 2003 challenge results.**

| Method (Team) | Score | BER | | AUC | | Fe | Pr |
|---|---|---|---|---|---|---|---|
| BayesNN-DFT (*Neal/Zhang*) | 71.4 | 6.48 | (1) | 97.20 | (1) | 80.3 | 47.8 |
| BayesNN-large (*Neal*) | 66.3 | 7.27 | (3) | 96.98 | (3) | 60.3 | 28.5 |
| BayesNN-small (*Neal*) | 61.1 | 7.13 | (2) | 97.08 | (2) | 4.7 | 2.9 |
| final_2-3 (*Chen*) | 49.1 | 7.91 | (8) | 91.45 | (25) | 24.9 | 9.9 |
| BayesNN-large (*Neal*) | 49.1 | 7.83 | (5) | 96.78 | (4) | 60.3 | 28.5 |
| final2-2 (*Chen*) | 40.0 | 8.80 | (17) | 89.84 | (29) | 24.6 | 6.7 |
| Ghostminer1 (*Ghostminer*) | 37.1 | 7.89 | (7) | 92.11 | (21) | 80.6 | 36.1 |
| RF+RLSC (*Torkkola/Tuv*) | 35.4 | 8.04 | (9) | 91.96 | (22) | 22.4 | 17.5 |
| Ghostminer2 (*Ghostminer*) | 35.4 | 7.86 | (6) | 92.14 | (20) | 80.6 | 36.1 |
| RF+RLSC (*Torkkola/Tuv*) | 34.3 | 8.23 | (12) | 91.77 | (23) | 22.4 | 17.5 |
| FS+SVM (*Lal*) | 31.4 | 8.99 | (19) | 91.01 | (27) | 20.9 | 17.3 |
| Ghostminer3 (*Ghostminer*) | 26.3 | 8.24 | (13) | 91.76 | (24) | 80.6 | 36.1 |
| CBAMethod3E (*CBAGroup*) | 21.1 | 8.14 | (10) | 96.62 | (5) | 12.8 | 0.1 |
| CBAMethod3E (*CBAGroup*) | 21.1 | 8.14 | (11) | 96.62 | (6) | 12.8 | 0.1 |
| Nameless (*Navot/Bachrach*) | 12.0 | 7.78 | (4) | 96.43 | (9) | 32.3 | 16.2 |

**Performance assessment**

Final submissions qualified for scoring if they included the class predictions for training, validation, and test sets for all five tasks proposed, and the list of features used. Optionally, classification confidence values could be provided. Performance was assessed using several metrics:

- BER: The balanced error rate, that is the average of the error rate of the positive class and the error rate of the negative class. This metric was used because some datasets (particularly DOROTHEA) are unbalanced.

- AUC: Area under the ROC curve. The ROC curve is obtained by varying a threshold on the discriminant values (outputs) of the classifier. The curve represents the fraction of true positive as a function of the fraction of false negative. For classifiers with binary outputs, BER=1-AUC.

- Ffeat: Fraction of features selected.
- Fprobe: Fraction of probes found in the feature set selected.

We ranked the participants with the test set results using a score combining BER, Ffeat and Fprobe. Briefly: We used the McNemar test to determine whether classifier A is better than classifier B according to the BER with 5% risk yielding to a score of 1 (better), 0 (don't know) or 1 (worse). Ties (zero score) were broken with Ffeat (if the relative difference in Ffeat was larger than 5%.) Remaining ties were broken with Fprobe. The overall score for each dataset is the sum of the pairwise comparison scores (normalized by the maximum achievable score, that is the number of submissions minus one.) The code is provided on the challenge website. The scores were averaged over the five datasets. Our scoring method favors accuracy over feature set compactness.

Our benchmark design could not prevent participants from "cheating" in the following way. An entrant could "declare" a smaller feature subset than the one used to make predictions. To deter participants from cheating, we warned them that we would be performing a stage of verification. We performed several checks as detailed in [9] and did not find any entry that should be suspected of being fraudulent.

## 3   Challenge results

The overall scores of the best entries are shown in Table 2. The main features of the methods of the participants listed in that table are summarized in Table 3. The analysis of this section also includes the survey of ten more top ranking participants.

**Winners**

The winners of the benchmark (both December $1^{st}$ and $8^{th}$) are Radford Neal and Jianguo Zhang, with a combination of Bayesian neural networks [14] and Dirichlet diffusion trees [15]. Their achievements are significant since they win on the overall ranking with respect to our scoring metric, but also with respect to the balanced error rate (BER), the area under the ROC curve (AUC), and they have the smallest feature set among the top entries that have performance not statistically significantly worse. They are also the top entrants December $1^{st}$ for ARCENE and DEXTER and December $1^{st}$ and $8^{th}$ for DOROTHEA.

Two aspects of their approach were the same for all data sets:

- They reduced the number of features used for classification to no more than a few hundred, either by selecting a subset of features using simple univariate significance tests, or by Principal Component Analysis (PCA) performed on all available labeled and unlabeled data.
- They then applied a classification method based on Bayesian learning, using an Automatic Relevance Determination (ARD) prior that allows the model to determine which of these features are most relevant.

Bayesian neural network learning with computation by Markov chain Monte Carlo (MCMC) is a well developed technology [14]. Dirichlet diffusion trees are a new Bayesian approach to density modeling and hierarchical clustering. As allowed by the challenge rules, the winners constructed these trees using both the training data and the unlabeled data in the validation and test sets. Classification was then performed with the k-nearest neighbors method, using the metric induced by the tree.

Table 3: **Methods employed by the challengers.** The classifiers are grouped in four categories: N - neural network, K - SVM or other kernel method, T - tree classifiers (none found in the top ranking methods), O - other. The feature selection engines (Fengine) are grouped in three categories: C - single variable criteria including correlation coefficients, T - tree classifiers or RF used as a filter E - Wrapper or embedded methods. The search methods are identified by: E - embedded, R - feature ranking, B - backward elimination, S - more elaborated search.

| Team | Classifier | Fengine | Fsearch | Ensemble | Transduction |
|------|-----------|---------|---------|----------|--------------|
| Neal/Zhang | N/O | C/E | E | Yes | Yes |
| Torkkola/Tuv | K | T | R | Yes | No |
| Chen/Lin | K | C/T/E | R/E | No | No |
| Zhili/Li | K | C/E | E | No | Yes |
| Saffari | N | C | R | Yes | No |
| Ghostminer | K | C/T | B | Yes | No |
| Lal et al | K | C | R | No | No |
| CBAGroup | K | C | R | No | No |
| Bachrach/Navot | K/O | E | S | No | No |

## Other methods employed

We group methods into coarse categories to draw useful conclusions. Our findings include:

**Feature selection** The winners and several top ranking challengers use a combination of filters and embedded methods[3]. Several high ranking participants obtain good results using only filters, even simple correlation coefficients. The second best entrants use Random Forests, an ensemble of tree classifiers, to perform feature selection [3].[4] Search strategies are generally unsophisticated (simple feature ranking, forward selection or backward elimination.) Only 2 out of 19 in our survey used a more sophisticated search strategy. The selection criterion is usually based on cross-validation. A majority use K-fold, with K between 3 and 10. Only one group used "random probes" purposely introduced to track the fraction of falsely selected features. One group used the area under the ROC curve computed on the training set.

**Classifier** Kernel methods [16] are most popular: 7/9 in Table 3 and 12/19 in the survey. Of the 12 kernel methods employed, 8 are SVMs. In spite of the high risk of overfitting, 7 of the 9 top groups using kernel methods found that Gaussian kernels gave them better results than the linear kernel on ARCENE, DEXTER, DOROTHEA, or GISETTE (for MADELON all best ranking groups used a Gaussian kernel.)

**Ensemble methods** Some groups relied on a committee of classifiers to make the final decision. The techniques to build such committee include sampling

from the posterior distribution using MCMC [14] and bagging [2]. Most groups that used ensemble methods reported improved accuracy.

**Transduction** Since all the datasets were provided since the beginning of the benchmark (validation and test set deprived of their class labels), it was possible to make use of the unlabelled data as part of learning (sometimes referred to as transduction [17]). Only two groups took advantage of that, including the winners.

**Preprocessing** Centering and scaling the features was the most common preprocessing used. Some methods required discretization of the features. One group normalized the patterns. Principal Componant Analysis (PCA) was used by several groups, including the winners, as a means of constructing features.

## 4   Conclusions and future work

The challenge demonstrated both that **feature selection can be performed effectively** and that **eliminating meaningless features is not critical to achieve good classification performance**. By design, our datasets include many irrelevant "distracters" features, called "probes". In contrast with redundant features, which may not be needed to improve accuracy but carry information, those distracters are "pure noise". It is surprising that some of the best entries use all the features. Still, there is always another entry close in performance, which uses only a small fraction of the original features.

The challenge outlined the power of filter methods. For many years, filter methods have dominated feature selection for computational reasons. It was understood that wrapper and embedded methods are more powerful, but too computationally expensive. Some of the top ranking entries use one or several filters as their only selection strategy. A filter as simple as the Pearson correlation coefficient proves to be very effective, even though it does not remove feature redundancy and therefore yields unnecessarily large feature subsets. Other entrants combined filters and embedded methods to further reduce the feature set and eliminate redundancies.

Another important outcome is that non-linear classifiers do not necessarily overfit. Several challenge datasets included a very large number of features (up to 100,000) and only a few hundred examples. Therefore, only methods that avoid overfitting can succeed in such adverse aspect ratios. Not surprisingly, the winning entries use as classifies either ensemble methods or strongly regularized classifiers. More surprisingly, non-linear classifiers often outperform linear classifiers. Hence, with adequate regularization, non-linear classifiers do not overfit the data, even when the number of features exceeds the number of examples by orders of magnitude.

Principal Component Analysis was successfully used by several researchers to reduce the dimension of input space down to a few hundred features, without any knowledge of the class labels. This was not harmful to the prediction performances and greatly reduced the computational load of the learning machines.

The analysis of the challenge results revealed that hyperparameter selection may have played an important role in winning the challenge. Indeed, several groups were using the same classifier (e.g. an SVM) and reported significantly different results. We have started laying the basis of a new benchmark on the theme of model selection and hyperparameter selection [8].

**Acknowledgments**

We are very thankful to the institutions that have contributed data: the National Cancer Institute (NCI), the Eastern Virginia Medical School (EVMS), the National Institute of Standards and Technology (NIST), DuPont Pharmaceuticals Research Laboratories, Reuters Ltd., and the Carnegie Group, Inc. We also thank the people who formatted the data and made them available: Thorsten Joachims, Yann Le Cun, and the KDD Cup 2001 organizers. We thank Olivier Chapelle for providing ideas and corrections. The workshop co-organizers and advisors Masoud Nikravesh, Kristin Bennett, Richard Caruana, and André Elisseeff, are gratefully acknowledged for their help, and advice, in particular with result dissemination.

## Footnotes

[1]After imposing a maximum of 5 submissions per group and eliminating some incomplete submissions, there remained 56 eligible submissions out of the 135 received.

[2]In this paper, we do not make a distinction between features and variables. The benchmark addresses the problem of selecting input variables. Those may actually be features derived from the original variables through preprocessing.

[3]We distinguish embedded methods that have a feature selection mechanism built into the learning algorithm from wrappers, which perform feature selection by using the classifier as a black box.

[4]Random Forests (RF) are classification techniques with an embedded feature selection mechanism. The participants used the features generated by RF, but did not use RF for classification.

# References

[1] A. Blum and P. Langley. Selection of relevant features and examples in machine learning. *Artificial Intelligence*, 97(1-2):245–271, December 1997.

[2] Leo Breiman. Bagging predictors. *Machine Learning*, 24(2):123–140, 1996.

[3] Leo Breiman. Random forests. *Machine Learning*, 45(1):5–32, 2001.

[4] S. Kremer, et al. NIPS 2000 unlabeled data competition. `http://q.cis.uoguelph.ca/~skremer/Research/NIPS2000/`, 2000.

[5] S. Kremer, et al. NIPS 2001 unlabeled data competition. `http://q.cis.uoguelph.ca/~skremer/Research/NIPS2001/`, 2001.

[6] I. Guyon. Design of experiments of the NIPS 2003 variable selection benchmark. `http://www.nipsfsc.ecs.soton.ac.uk/papers/Datasets.pdf`, 2003.

[7] I. Guyon and A. Elisseeff. An introduction to variable and feature selection. *JMLR*, 3:1157–1182, March 2003.

[8] I. Guyon and S. Gunn. Model selection and ensemble methods challenge in preparation `http://clopinet.com/isabelle/projects/modelselect`.

[9] I. Guyon, S. Gunn, M. Nikravesh, and L. Zadeh, Editors. *Feature Extraction, Foundations and Applications.* Springer-Verlag, `http://clopinet.com/isabelle/Projects/NIPS2003/call-for-papers.html`, In preparation. See also on-line supplementary material: `http://clopinet.com/isabelle/Projects/NIPS2003/analysis.html`.

[10] D. Kazakov, L. Popelinsky, and O. Stepankova. MLnet machine learning network on-line information service. In `http://www.mlnet.org`.

[11] R. Kohavi and G. John. Wrappers for feature selection. *Artificial Intelligence*, 97(1-2):273–324, December 1997.

[12] D. LaLoudouana and M. Bonouliqui Tarare. Data set selection. In *NIPS02* `http://www.jmlg.org/papers/laloudouana03.pdf`, 2002.

[13] P. M. Murphy and D. W. Aha. UCI repository of machine learning databases. In `http://www.ics.uci.edu/~mlearn/MLRepository.html`, 1994.

[14] R. M. Neal. *Bayesian Learning for Neural Networks.* Number 118 in Lecture Notes in Statistics. Springer-Verlag, New York, 1996.

[15] R. M. Neal. Defining priors for distributions using dirichlet diffusion trees. Technical Report 0104, Dept. of Statistics, University of Toronto, March 2001.

[16] B. Schoelkopf and A. Smola. *Learning with Kernels – Support Vector Machines, Regularization, Optimization and Beyond.* MIT Press, Cambridge MA, 2002.

[17] V. Vapnik. *Statistical Learning Theory.* John Wiley &amp; Sons, N.Y., 1998.